# An Efficient Clustering Algorithm Using Stochastic Association Model and Its Implementation Using Nanostructures

**Takashi Morie, Tomohiro Matsuura, Makoto Nagata, and Atsushi Iwata**
Graduate School of Advanced Sciences of Matter, Hiroshima University
Higashi-hiroshima, 739-8526 Japan.
`http://www.dsl.hiroshima-u.ac.jp`
*morie@dsl.hiroshima-u.ac.jp*

## Abstract

This paper describes a clustering algorithm for vector quantizers using a "stochastic association model". It offers a new simple and powerful soft-max adaptation rule. The adaptation process is the same as the on-line K-means clustering method except for adding random fluctuation in the distortion error evaluation process. Simulation results demonstrate that the new algorithm can achieve efficient adaptation as high as the "neural gas" algorithm, which is reported as one of the most efficient clustering methods. It is a key to add uncorrelated random fluctuation in the similarity evaluation process for each reference vector. For hardware implementation of this process, we propose a nanostructure, whose operation is described by a single-electron circuit. It positively uses fluctuation in quantum mechanical tunneling processes.

## 1 Introduction

Vector quantization (VQ) techniques are used in a wide range of applications, including speech and image processing, data compression. VQ techniques encode a data manifold $V \subseteq \Re^D$ using only a finite set of reference vectors $\boldsymbol{w} = (\boldsymbol{w}_1, \cdots, \boldsymbol{w}_N)$. A data vector $\boldsymbol{v} \in V$ is represented by the best-matching or "winning" reference vector $\boldsymbol{w}_c$, which minimizes the average distortion error:

$$E = \int \|\boldsymbol{v} - \boldsymbol{w}_c\|^2 p(\boldsymbol{v}) d^D v, \tag{1}$$

where $p(\boldsymbol{v})$ is the probability distribution of data vectors over manifold $V$.

Various clustering algorithms to obtain the best reference vectors have been reported. Here, we treat on-line training, in which the data point distribution is not given a priori, but instead a stochastic sequence of incoming sample data points drives the adaptation procedure.

The straightforward approach is the well-known *on-line K-means clustering* algorithm, in which only the nearest reference vector to the sample vector is adjusted;

$$\Delta \boldsymbol{w}_i = \varepsilon \cdot \delta_{ic} \cdot (\boldsymbol{v}(t) - \boldsymbol{w}_i), \tag{2}$$

where, $\varepsilon$ is the step size and $\delta_{ij}$ is the Kronecker delta. However, this simple clustering algorithm is often stuck in a local minimum. To avoid this difficulty, a common approach is to introduce a "soft-max" adaptation rule that not only adjusts the "winning" reference vector but affects other reference vectors depending on their proximity to $v$.

The *maximum-entropy* (ME) algorithm [1] adjusts all reference vectors $w_i$ depending on the Euclidean distance to $v$;

$$\Delta w_i = \varepsilon \cdot \frac{e^{-\beta(v-w_i)^2}}{\sum_{j=1}^{N} e^{-\beta(v-w_j)^2}} \cdot (v(t) - w_i), \tag{3}$$

where parameter $\beta$ defines the proximity.

The *Kohonen's self-organization map* (SOM) algorithm [2] is another well-known model;

$$\Delta w_i = \varepsilon \cdot h_\sigma(i, c) \cdot (v(t) - w_i). \tag{4}$$

In this model, every reference vector is assigned to a site of a lattice. Each time a sample vector is presented, not only the "winning" reference vector is adjusted but also the reference vectors assigned to the lattice sites adjacent to the winner are updated according to function $h_\sigma(i, c)$, which is typically chosen to be a Gaussian:

$$h_\sigma(i, c) = e^{-\|w_i - w_c\|^2/(2\sigma^2)}, \tag{5}$$

where $\sigma$ is a parameter that defines the proximity.

The *neural-gas* (NG) clustering algorithm [3] is a powerful soft-max adaptation rule, in which all reference vectors are adjusted depending on the "neighborhood ranking";

$$\Delta w_i = \varepsilon \cdot h_\lambda(k_i(v, w)) \cdot (v(t) - w_i), \tag{6}$$

where $k_i(v, w)$ is the ranking, which depends on $v$ and the whole set $w$. The function $h_\lambda(k_i)$ is typically as follows:

$$h_\lambda(k) = e^{-k/\lambda}, \tag{7}$$

where parameter $\lambda$ defines the proximity. This algorithm exhibits faster convergence to smaller distortion errors, however consumes higher computational power especially for sorting. An efficient version of the NG clustering that adjusts only several reference vectors having upper ranking was also proposed [4].

In the next section, we propose a new efficient soft-max adaptation algorithm. It employs the *stochastic association model* that we have proposed related to single-electron circuits [5], [6]. In Sec. 3, it is demonstrated from simulation results that this new clustering algorithm is as powerful as the other algorithms. In Sec. 4, we propose a nanostructure based on a single-electron circuit for implementing the stochastic association model.

## 2   Stochastic association algorithm

A usual associative memory is defined as a system that *deterministically* extracts the vector most similar to the input vector from the stored reference vectors. This just corresponds to the process choosing the winning reference vector for a certain data vector in all conventional clustering algorithms.

In our stochastic association (SA) model, the association probability depends on the similarity between the input and the reference vectors. The SA algorithm extracts not only the reference vector most similar to the input but also other similar reference vectors with the probability depending on the similarity.

In the SA algorithm, stochastic fluctuation is added in the evaluation process of distortion error $D_i$ between data vector $v$ and reference vector $w_i$. We propose this algorithm inspired

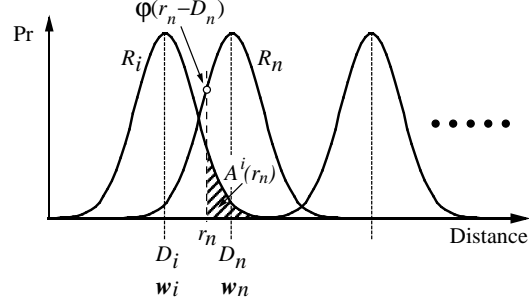

Figure 1: Probability distribution in evaluation of the distortion error between the data vector and each reference vector.

by the quantum mechanical property of single-electron circuits as described in Sec. 4, and we expect that such fluctuation helps to avoid getting stuck in local minima of $E$.

The distortion error $D_i$ can be the squared Euclidean distance $\|v - w_i\|^2$ or the Manhattan distance $\|v - w_i\|$. The evaluation result is represented by

$$R_i = D_i + \xi, \tag{8}$$

where $\xi$ is a random variable with probability distribution function $\varphi(\xi)$. Therefore, the evaluation result $R_i$ is also considered as a random variable. The probability that $R_i$ has value $r_i$ is represented by

$$\Pr\{R_i = r_i\} = \varphi(r_i - D_i). \tag{9}$$

The winning reference vector $w_c$ is determined by

$$c = \arg\min_i\{R_i\}. \tag{10}$$

The probability that reference vector $w_n$ becomes the winner when $R_n$ has value $r_n$ for a certain data vector is given by the product of $\varphi(r_n - D_n)$ and the probability that $R_i > r_n, \forall i \neq n$ as shown in Fig. 1. Therefore, the probability that $w_n$ becomes the winner is given by integrating it with $r_n$;

$$\Pr\{c = n\} = \int_{-\infty}^{\infty} dr_n \varphi(r_n - D_n) \prod_{i \neq n} A^i(r_n) \tag{11}$$

$$A^i(r_n) \equiv \int_{r_n}^{\infty} \varphi(r - D_i)dr. \tag{12}$$

If the winning reference vector is updated as expressed by eq. (2), the SA model can provide a new soft-max adaptation rule. Figure 2 shows an architecture for clustering processing using the SA model. The distortion error between the input vector and each stored reference vector is evaluated in parallel with stochastic fluctuation. The winner-take-all circuit deterministically extracts the winner, and the winning reference vector is only updated with a constant value. As in the K-means algorithm, only one reference vector is adjusted for each adaptation step and the update value for the selected reference vector is independent of similarity or proximity. However, unlike the K-means algorithm, the adjusted vector is not always the most similar reference vector, and sometimes other similar vectors are adjusted. The total adjusting tendency in the SA algorithm seems similar to the NG or ME algorithm because the probability of reference vector selection is determined by the neighborhood ranking and the distances between each reference vector and a given data vector.

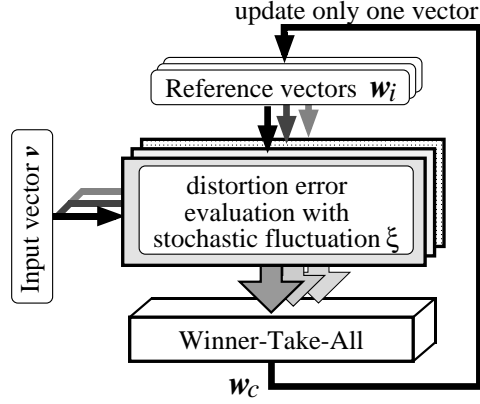

Figure 2: Architecture for clustering processing using the SA model.

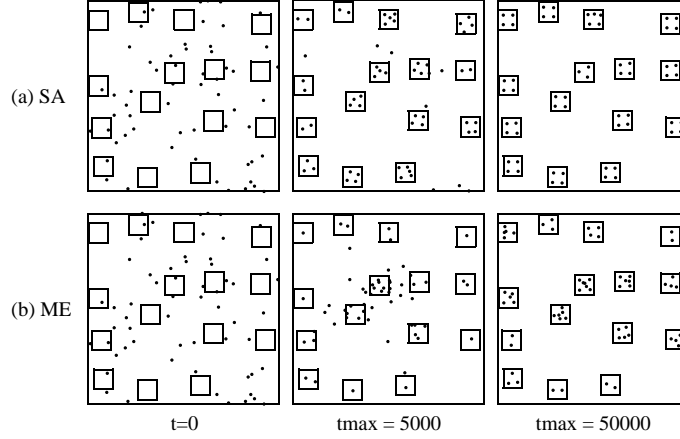

Figure 3: Test problem and clustering results by SA and ME algorithms. Data samples uniformly distribute in square regions, and points represent reference vectors. Both algorithms use the same initial state.

## 3  Simulation results

In order to test the performance of the SA algorithm in minimizing the distortion error and to compare it with the other soft-max approaches, we performed the same simulation of model clustering described by Ref. [3]. The data clusters are of square shape within a two-dimensional input space as shown in Fig. 3. In the simulation, the number of clusters was 15, and that of reference vectors was 60. We averaged the results of 50 simulation runs for each of which not only the initialization of the reference vectors were chosen randomly but also the 15 clusters were placed randomly.

The SA algorithm in this simulation used the squared Euclidean distance as a distortion error $D_i$ and the *normal distribution* as the probability distribution of the stochastic fluctuation;

$$\varphi(\xi) = N(0, \gamma^2) \equiv \frac{1}{\sqrt{2\pi}\gamma} \exp\left(-\frac{\xi^2}{2\gamma^2}\right). \tag{13}$$

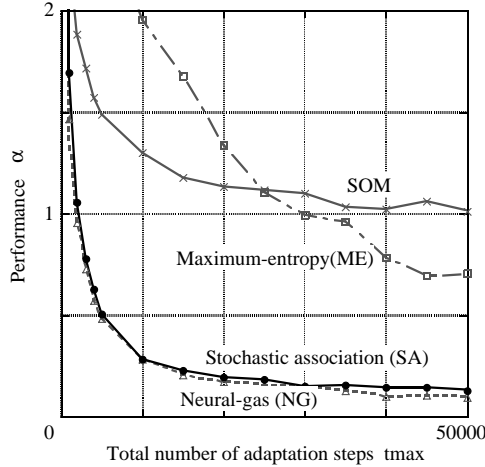

| algorithm | parameter | initial $x_i$ | final $x_f$ |
|:---:|:---:|:---:|:---:|
| ME | $\beta$ | 1 | 10000 |
| SOM | $\sigma$ | 2 | 0.01 |
| NG | $\lambda$ | 10 | 0.01 |
| SA | $\gamma$ | 0.2 | 0.0001 |
| All | $\varepsilon$ | 0.5 | 0.005 |

Figure 4: Clustering performance of SA algorithm comparing with other clustering methods. The optimized parameters used in the simulation are also shown.

Figure 3 shows an example of clustering by the SA algorithm compared with that by the ME algorithm. The result of the SA algorithm demonstrates nearly perfect clustering for $t_{max} = 50000$. In contrast, the clustering result by the ME algorithm is not so good although the parameters used were optimized.

Here, all the clustering algorithms including the SA algorithm use an annealing procedure to escape local minima. The parameters were gradually reduced during adaptation:

$$x(t) = x_i (x_f / x_i)^{t/t_{max}} \qquad \text{for } \beta, \sigma, \lambda, \gamma, \varepsilon \qquad (14)$$

where $t_{max}$ is the total number of adaptation steps. The values optimized by numerous preliminary simulations are shown in Fig. 4, which were used in the simulation described here.

In order to compare the performance of the algorithms, we used a performance measure $\alpha = [E(t_{max}) - E_0]/E_0$, where $E_0$ is the minimal distortion error in this problem. The relationships between $t_{max}$ and $\alpha$ for the four algorithms are shown in Fig. 4. The clustering performance of the SA algorithm is nearly equal to that of the NG algorithm, which is the most efficient clustering method in this test problem. The number of adaptation steps to reach the steady state and the distortion error at the steady state in the SA algorithm are nearly the same as those in the NG algorithm.

We also performed other simulations, one of which was vector quantization of a real image ('Lena', $256 \times 256$ pixels, 8-bit grayscale). In this case, the SOM demonstrated the best performance, and the SA algorithm also had the nearly equal performance.

Consequently, comparing with the other soft-max algorithms, the SA algorithm has nearly the best clustering performance. Moreover, it does not require a sorting process unlike the NG algorithm nor a searching process of adjacent lattice sites unlike the SOM; only one reference vector is adjusted per adaptation step. Thus, the computational power required by the SA algorithm is much less than that required by the other soft-max algorithm. If the number of reference vectors is $N$, the total updating steps of reference vectors in the SA algorithm are $1/N$ times as many as those in the other algorithms. Thus, the SA algorithm is the most efficient clustering method.

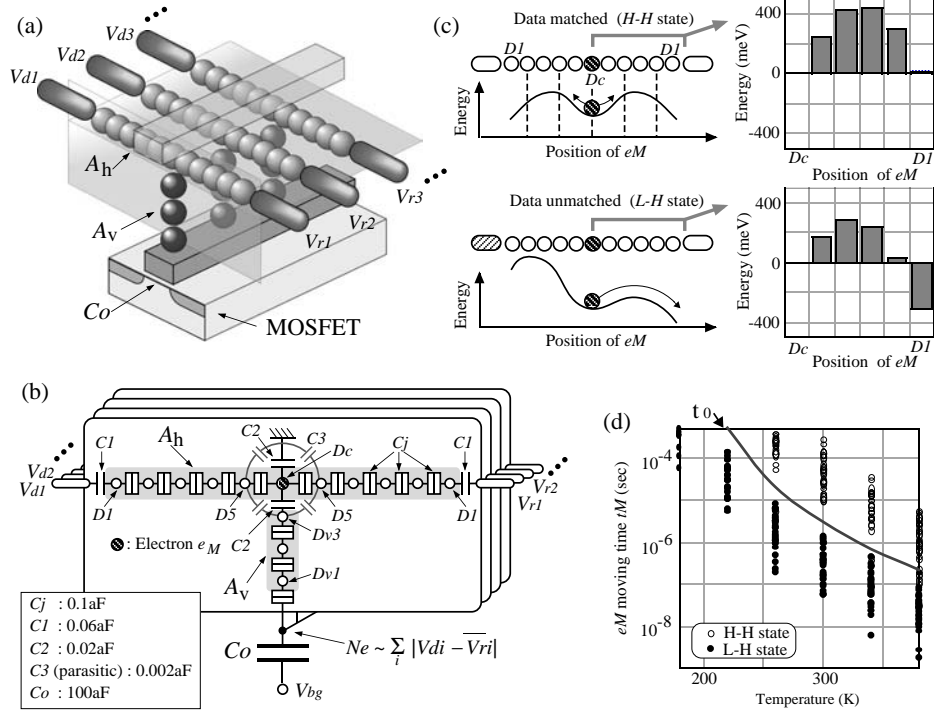

Figure 5: Nanostructure evaluating Hamming distance. (a) Schematic of nanostructure, where dot arrays are extremely enlarged compared with a MOSFET to emphasize the dot structures. (b) Single-electron circuit. (c) Potential profile in dot array $A_h$. (d) $e_M$ moving time for bit comparator operation.

## 4  Nanostructure implementing SA model

The key for implementing the SA model is adding random fluctuation as expressed by eq. (8). We have already proposed single-electron circuits and nanostructures evaluating Hamming distance for the SA model [5]-[9].

Figure 5(a) and (b) show a nanostructure and the corresponding single-electron circuit, respectively, which are the most sophisticated version of our circuits and structures [9]. The nanostructure consists of plural ($M$) dot structures arranged on a MOS transistor gate electrode. Each dot structure consists of 1-D dot arrays $A_h$ ($D_1, \cdots, D_n, D_c, D_n, \cdots, D_1$) and $A_v$ ($D_{v1}, D_{v2}, D_{v3}$), where $n$ means the number of dots at a side of $A_h$. (From Monte Carlo single-electron circuit simulation, $n$ should be more than 3). The dot diameter assumed is around 1 nm. The capacitance $C_o$ corresponds to the gate capacitance of an ultrasmall MOS transistor. An electron $e_M$ is introduced in array $A_h$, which is for example performed by using Fowler-Nordheim tunneling from the grounded plate over $D_c$. Electron $e_M$, which is initially located at $D_c$, can move along array $A_h$ through tunneling junctions $C_j$, but it cannot move to $A_v$ through the normal capacitor $C_2$. Digital (High/Low) voltages $V_{di}$ and $V_{ri}$ ($i = 1, 2, \cdots, M$) are applied at both edges of $A_h$, which correspond to elements of data and reference vectors, respectively. Each dot structure simply works as an exclusive-NOR logic gate (bit comparator) with random fluctuation as explained below.

If the two digital data bits ($H$ or $L$) are matched, electron $e_M$ stabilizes at center dot $D_c$, otherwise $e_M$ moves to an off-center position. After stabilizing $e_M$, by changing voltages

$V_{di}$, $V_{ri}$ and back-gate voltage $V_{bg}$, vertical dot array $A_v$ detects whether $e_M$ stays at $D_c$ or not; only if $e_M$ stays at $D_c$, $A_v$ is polarized and an electron is induced at the gate electrode of $C_o$. The total number of induced electrons ($N_e$) is proportional to the number of dot structures with matched bits; thus the Hamming distance can be measured by counting the induced electrons using the ultrasmall MOS transistor. (If one of the input digital data is applied through an inverter, the number of unmatched bits can be calculated).

The detail of operation stabilizing $e_M$ is as follows: Because of the charging energy of $e_M$ itself, the total energy as a function of the position of $e_M$ in array $A_h$ has two peaks at the midpoints of each side of the array, and has minimal values at $D_c$ and both of $D_1$ as shown in Fig. 5(c). The energy barrier height for $e_M$ at $D_c$ is assumed larger than the thermal energy at room temperature.

In *L-L state*, the energy at $D_1$ rises up, thus $e_M$ is most strongly stabilized at $D_c$. On the other hand, in *H-L(L-H)* or *H-H state*, the energy barrier is lower than that of *L-L state*, thus $e_M$ can more easily overcome the barrier by using thermal noise. Figure 5(d) shows the relation between operation temperature and time ($t_M$) required until $e_M$ moves to $D_1$, which was obtained by Monte Carlo single-electron circuit simulation. The moving process assisted by thermal noise is purely stochastic, thus $t_M$ scatters in a wide range. However, because the energy barrier height in *H-L(L-H) states* is lower than that in *H-H state* as shown in Fig. 5(c), there exists a certain time span $t_0$ within which $e_M$ in *H-L(L-H) states* moves to $D_1$ while $e_M$ in *H-H state* stays at $D_c$. At room temperature (300K), $t_0$ is several microseconds in this case although $t_0$ depends on the tunneling resistance. If the detection process starts after $t_0$, nearly perfect exclusive-NOR (bit comparison) operation is achieved. On the other hand, if the start timing is shifted from $t_0$, arbitrary amount of fluctuation can be included in the bit comparison result. Thus, we utilize quantum mechanical tunneling processes assisted by thermal noise in this structure, which is similar to a phenomenon known as *stochastic resonance*.

Although digital data are treated in the above explanation, analog data can be treated in the same circuit by using pulse-width modulation (PWM) signals, which have a digital amplitude and an analog pulse width [10]. Therefore, instead of the Hamming distance, the Manhattan distance can be evaluated by using this nanostructure. Because random fluctuation is naturally added in our nanostructure, it can implement the calculation expressed by eq. (8). The annealing procedure described by eqs. (13) and (14) can be performed by changing the time scale in the stabilization operation; that means the scaling of pulse-width modulation.

The proposed nanostructure has not yet been fabricated using the present VLSI technology, but the basic technology related to nanocrystalline floating-dot MOSFET devices, which are closely related to our structure, is now being developed [11]-[13]. Furthermore, well-controlled self-assembly processes using molecular manipulation technology, especially using DNA [14], would be utilized to fabricate our nanostructure. Thus, it could be constructed in the near future.

## 5   Conclusions

The stochastic association algorithm offers a simple and powerful soft-max adaptation rule for vector quantizers. Although it is the same as the simple on-line K-means clustering method except for adding random fluctuation in the distortion error evaluation process, our new method has an efficient adaptation performance as high as the neural-gas (NG) or the SOM algorithms. Moreover, our method needs no additional process such as sorting and only one reference vector is adjusted at each adaptation step; thus the computational effort is much smaller compared with the conventional soft-max clustering algorithms.

By employing the nanostructure proposed in this paper, very high performance clustering hardware could be constructed.

## Acknowledgments

The authors wish to thank Prof. Masataka Hirose for his support and encouragement. This work has been supported in part by Grants-in-aid for the Core Research for Evolutional Science and Technology (CREST) from Japan Science and Technology Corporation(JST).

## References

[1] K. Rose, E. Gurewitz, and G. C. Fox, "Statistical Mechanics and Phase Transitions in Clustering," *Physical Review Letters*, vol. 65, no. 8, pp. 945–948, 1990.

[2] T. Kohonen, *Self-Organization and Associative Memory*, Springer-Verlag, Berlin, 1984.

[3] T. M. Martinetz, S. G. Berkovich, and K. J. Schulten, "'Neural-Gas" Network for Vector Quantization and its Apllication to Time-Series Prediction," *IEEE Trans. Neural Networks*, vol. 4, pp. 558–569, 1993.

[4] S. Rovetta and R. Zunino, "Efficient Training of Neural Gas Vector Quantizers with Analog Circuit Implementation," *IEEE Trans. Circuits & Syst.*, vol. 46, pp. 688–698, 1999.

[5] M. Saen, T. Morie, M. Nagata, and A. Iwata, "A Stochastic Associative Memory Using Single-Electron Tunneling Devices," *IEICE Trans. Electron.*, vol. E81-C, no. 1, pp. 30–35, 1998.

[6] T. Yamanaka, T. Morie, M. Nagata, and A. Iwata, "A Single-Electron Stochastic Associative Processing Circuit Robust to Random Background-Charge Effects and Its Structure Using Nanocrystal Floating-Gate Transistors," *Nanotechnology*, vol. 11, no. 3, pp. 154–160, 2000.

[7] T. Morie, T. Matsuura, S. Miyata, T. Yamanaka, M. Nagata, and A. Iwata, "Quantum Dot Structures Measuring Hamming Distance for Associative Memories," *Superlattices & Microstructures*, vol. 27, no. 5/6, pp. 613–616, 2000.

[8] T. Matsuura, T. Morie, M. Nagata, and A. Iwata, "A Multi-Quantum-Dot Associative Circuit Using Thermal-Noise Assisted Tunneling," in *Ext. Abs. of Int. Conf. on Solid State Devices and Materials*, pp. 306–307, Sendai, Japan, Aug. 2000.

[9] T. Morie, T. Matsuura, M. Nagata, and A. Iwata, "Quantum Dot Structures Measuring Hamming Distance for Associative Memories," in *Extended Abstracts, 4th International Workshop on Quantum Functional Devices (QFD2000)*, pp. 210–213, Kanazawa, Japan, Nov. 2000.

[10] A. Iwata and M. Nagata, "A Concept of Analog-Digital Merged Circuit Architecture for Future VLSI's," *IEICE Trans. Fundamentals.*, vol. E79-A, no. 2, pp. 145–157, 1996.

[11] S. Tiwari, F. Rana, H. Hanafi ,A. Hartstein, E. F. Crabbé, and K. Chan, "A Silicon Nanocrystals Based Memory," *Appl. Phys. Lett.*, vol. 68, no. 10, pp. 1377–1379, 1996.

[12] A. Kohno, H. Murakami, M. Ikeda, H. Nishiyama, S. Miyazaki, and M. Hirose, "Transient Characteristics of Electron Charging in Si-Quantum-Dot Floating Gate MOS Memories," in *Ext. Abs. of Int. Conf. on Solid State Devices and Materials*, pp. 124–125, Sendai, Japan, Aug. 2000.

[13] R. Ohba, N. Sugiyama, J. Koga, K. Uchida, and A. Toriumi, "Novel Si Quantum Memory Structure with Self-Alighed Stacked Nanocrystalline Dots," in *Ext. Abs. of Int. Conf. on Solid State Devices and Materials*, pp. 122–123, Sendai, Japan, Aug. 2000.

[14] R. A. Kiehl, "Nanoelectronic Array Architecture," in *Extended Abstracts, 4th International Workshop on Quantum Functional Devices (QFD2000)*, pp. 49–51, Kanazawa, Japan, Nov. 2000.